# Chaitin-Kolmogorov Complexity and Generalization in Neural Networks

**Barak A. Pearlmutter**
School of Computer Science
Carnegie Mellon University
Pittsburgh, PA 15213

**Ronald Rosenfeld**
School of Computer Science
Carnegie Mellon University
Pittsburgh, PA 15213

## Abstract

We present a unified framework for a number of different ways of failing to generalize properly. During learning, sources of random information contaminate the network, effectively augmenting the training data with random information. The complexity of the function computed is therefore increased, and generalization is degraded. We analyze replicated networks, in which a number of identical networks are independently trained on the same data and their results averaged. We conclude that replication almost always results in a decrease in the expected complexity of the network, and that replication therefore increases expected generalization. Simulations confirming the effect are also presented.

## 1 BROKEN SYMMETRY CONSIDERED HARMFUL

Consider a one-unit backpropagation network trained on exclusive or. Without hidden units, the problem is insoluble. One point where learning would stop is when all weights are zero and the output is always $\frac{1}{2}$, resulting in an mean squared error of $\frac{1}{4}$. But this is a saddle point; by placing the discrimination boundary properly, one point can be gotten correctly, two with errors of $\frac{1}{3}$, and one with error of $\frac{2}{3}$, giving an MSE of $\frac{1}{6}$, as shown in figure 1.

Networks are initialized with small random weights, or noise is injected during training to break symmetries of this sort. But in breaking this symmetry, something has been lost. Consider a $k$NN classifier, constructed from a $k$NN program and the training data. Anyone who has a copy of the $k$NN program can construct an *identical* classifier if they receive the training data. Thus, considering the classification

as an abstract entity, we know its complexity cannot exceed that of the training data plus the overhead of the complexity of the program, which is fixed.

But this is not necessarily the case for the backpropagation network we saw! Because of the introduction of randomly broken symmetries, the complexity of the classification itself can exceed that of the training data plus the learning procedure. Thus an identical classifier can no longer be constructed just from the program and the training data, because random factors have been introduced. For a striking example, consider presenting a "32 bit parity with 10,000 exceptions" stochastic learner with one million exemplars. The complexity of the resulting function will be high, since in order to specify it we must specify not only the regularities of training set, which we just did in a couple words, but also which of the 4 billion possibilities are among the 10,000 exceptions.

Applying this idea to undertraining and overtraining, we see that there are two kinds of symmetries that can be broken. First, if not all the exemplars can be loaded, which of the outliers are not loaded can be arbitrary. Second, underconstrained networks that behave the same on the training set may behave differently on other inputs. Both phenomena can be present simultaneously.

## 2    A COMPLEXITY BOUND

The expected value of the complexity of the function implemented by a network $b$ trained on data $d$, where $b$ is a potentially stochastic mapping, satisfies

$$E(\mathcal{C}(b(d))) \leq \mathcal{C}(d) + \mathcal{C}(b) + I(b(d)|d)$$

where $I(b(d)|d)$ is the negative of the entropy of the bias distribution of $b$ trained on $d$,

$$I(b(d)|d) = -H(b(d)) = -\sum_f \log P(b(d) = f)$$

where $f$ ranges over *functions* that the network could end up performing, with the network regarded as a black box. This in turn is bounded by the information contained in the random internal parameters, or by the entropy of the watershed structure; but these are both potentially unbounded.

A number of techniques for improving generalization, when viewed in this light, work because they tighten this bound.

- Weight decay [2] and the statistical technique of ridge regression impose an extra constraint on the parameters, reducing their freedom to arbitrarily break symmetry when underconstrained.

- Cross validation attempts to stop training before too many symmetries have been broken.

- Efforts to find the perfect number of hidden units attempt to minimize the number of symmetries that must be broken.

These techniques strike a balance between undertraining and overtraining. Since in any realistic domain both of these effects will be simultaneously present, it would seem advantageous to attack the problem at the root. One approach that has been

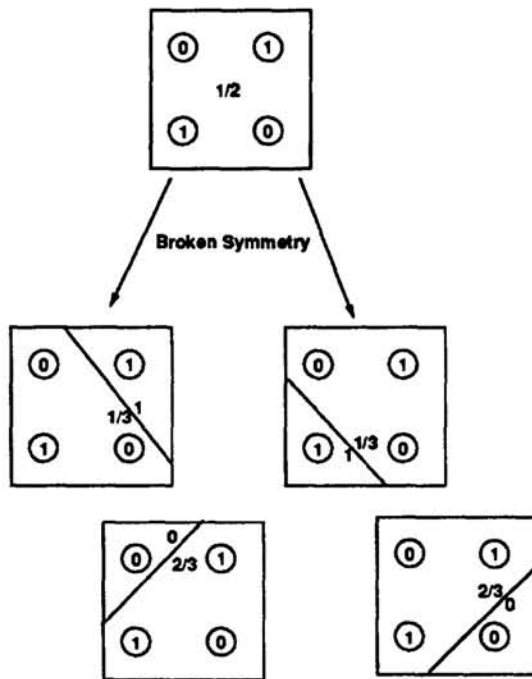

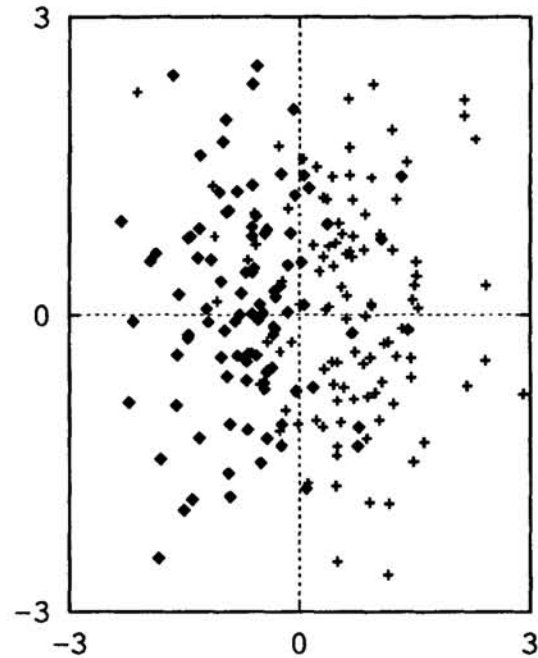

Figure 1: The bifurcation of a perceptron trained on xor.

Figure 2: The training set. Crosses are negative examples and diamonds are positive examples.

rediscovered a number of times [1, 3], and systematically explored in its pure form by Lincoln and Skrzypek [4], is that of replicated networks.

## 3  REPLICATED NETWORKS

One might think that the complexity of the average of a collection of networks would be the sum of the complexities of the components; but this need not be the case. Consider an ensemble network, in which an infinite number of networks are taught the training data simultaneously, each making its random decisions according to whatever distributions the training procedure calls for, and their output averaged.

We have seen that the complexity of a single network can exceed that of its training data plus the training program. But this is not the case with ensemble networks, since the ensemble network output can be determined solely from the program and the training data, i.e. $C(E(b(d))) \leq C(b) + C(d) + C(\text{"replicate"})$ where $C(\text{"replicate"})$ is the complexity of the instruction to replicate and average (a small constant).

A simple way to approximate the ensemble machine is to train a number of networks simultaneously  and average the results. As the number of networks is increased, the composite model approaches the ensemble network, which  cannot have higher complexity than the training data plus the program plus the instruction to replicate.

Note that even if one accidentally stumbles across the perfect architecture and

training regime, resulting in a net that always learns the training set perfectly but with no leftover capacity, and which generalizes as well as anything could, then making a replicated network can't hurt, since all the component networks would do exactly the same thing anyway.

A number of researchers seem to have inadvertently exploited this fact. For instance, Hampshire et al. [1] train a number of networks on a speech task, where the networks differed in choice of objective function. The networks' outputs were averaged to form the answer used in the recognition phase, and the generalization performance of the composite network was significantly higher than that of any of its component networks. Replicated implementations programmed from identical specifications is a common technique in software engineering of highly reliable systems.

## 4    THE ISSUE OF INDUCTIVE BIAS

The representational power of an ensemble is greater that that of a single network. By the usual logic, one would expect the ensemble to have worse generalization, since its inductive bias is weaker. Counterintuitively, this is not the case. For instance, the VC dimension of an ensemble of perceptrons is infinite, because it can implement an arbitrary three layer network, using replication to implement weights. This is much greater than the finite VC dimension of a single perceptron within the ensemble, but our analysis predicts better generalization for the ensemble than for a single stochastic perceptron when the bounds are tight, that is, when

$$H(b(d)) \geq \mathcal{C}(\text{"replicate"}). \tag{1}$$

This leads to the conclusion that just knowing the inductive bias of a learner is not enough information to make strong conclusions about its expected generalization. Thus, distribution free results based purely on the inductive bias, such as VC dimension based PAC learning theory [5], may sometimes be unduly pessimistic.

As to replicated networks, we have seen that they can not help but improve generalization when (1) holds. Thus, if one is training the same network over and over, perhaps with slightly different training regimes, and getting worse generalization than was hoped for, but on different cases each time, then one can improve generalization in a seemingly principled manner by putting all the trained networks in a box and calling it a finite sample of the ensemble network (and perhaps buying a bigger computer to run it on).

## 5    EMPIRICAL SUPPORT

We conducted the following experiment: 17 standard backpropagation networks (Actually 20, but 3 were lost to a disk failure) were trained on a binary classification task. The nets all had identical architectures (2–20–1) but different initial weights, chosen uniformly from the interval $[-1, 1]$. The same training set was used to train all the networks. The functions implemented by each of the networks were then calculated in detail, and the performance of individual networks compared to that of their ensemble.

The classification task was a stochastic 2D linear discriminator. Each point was obtained from a Gaussian centered at (0.0) with stdev 1. A classification of 1 was

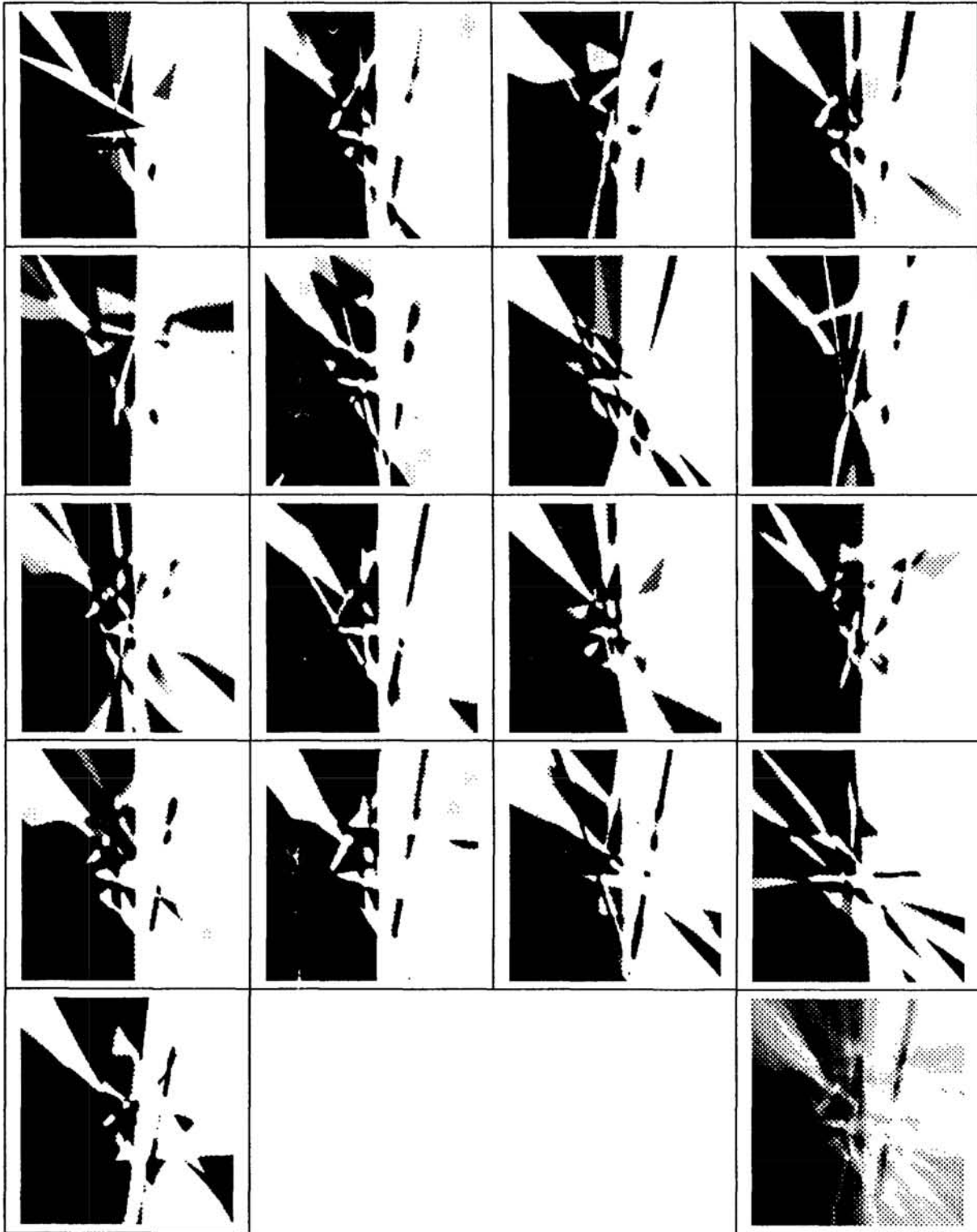

Figure 3: The functions implemented by the 17 trained networks, and by their average (bottom right). Both the x and y axes run from -3 to 3, and grey levels are used to represent intermediate values in the interval [0, 1].

Table 1: Mean squared error and number of mislabeled exemplars for each network on the training set of 200.

| net | MSE | errors | |
|---|---|---|---|
| 12 | 0.0150837 | 3 | *** |
| 9 | 0.0200039 | 4 | **** |
| 16 | 0.0200026 | 4 | **** |
| 5 | 0.0250207 | 5 | ***** |
| 7 | 0.0250213 | 5 | ***** |
| 10 | 0.0228319 | 5 | ***** |
| 13 | 0.0250156 | 5 | ***** |
| 17 | 0.0250018 | 5 | ***** |
| 19 | 0.0175466 | 5 | ***** |
| 6 | 0.0300099 | 6 | ****** |
| 15 | 0.0300075 | 6 | ****** |
| 18 | 0.0300060 | 6 | ****** |
| 8 | 0.0350609 | 7 | ******* |
| 11 | 0.0350006 | 7 | ******* |
| 20 | 0.0400013 | 8 | ******** |
| 14 | 0.0305254 | 9 | ********* |
| 4 | 0.0408391 | 13 | ************* |
| mean | 0.027469 ± 0.007226 | 6.058824 ± 2.261457 | |
| ensemble | 0.016286 | 4 | **** |
| nohidden | 0.060314 | 31 | |

assigned to points with $x \geq 0$, and 0 to points with $x < 0$, but reversed with an independent probability of 0.1. The final position of each point was then determined by adding a zero mean Gaussian with stdev .25. 200 points were so generated for the training set (shown in figure 2) and another 1000 points for the test set.

Looking at figure 3, each net appears to correctly classify as many of the inputs as possible, within the bounds imposed on it by its inductive bias. Each function implemented by such a net is roughly equivalent to a linear combination of 20 independent linear discriminators. It is therefore clear why each map consists of regions delineated by up to 20 straight lines. Since the initial conditions were different for each net, so were the resultant regions. All networks misclassified some of the exemplars (see table 1), but the missclassifications were different for each network, illustrating symmetry breaking due to an overconstraining data set.

Note that the ensemble's performance on the training set is comparable to that of the best of the trained networks, while its performance on the test set is far superior. The MSE error of the ensemble is much much better than the bound obtained from Jensen's inequality, the average MSE. In fact, the ensemble network gets a lower MSE than all but one individual network on the training sets, and a much lower MSE than any individual network on the test set; and it generalizes much better than any of the individual networks by a misclassification count metric.

Table 2: Mean squared error and number of mislabeled samples for each network on the test set of 1000. The performance of a theoretically perfect classifier (sign $x$) on the test set is 170 misclassifications, which is about what the network without hidden units gets.

| net | MSE | errors |
|-----|-----|--------|
| 16 | 0.201 | 205 |
| 9 | 0.207 | 213 |
| 4 | 0.206 | 215 |
| 5 | 0.209 | 216 |
| 11 | 0.208 | 216 |
| 15 | 0.207 | 216 |
| 6 | 0.212 | 219 |
| 19 | 0.213 | 220 |
| 7 | 0.214 | 222 |
| 8 | 0.214 | 224 |
| 12 | 0.212 | 225 |
| 17 | 0.219 | 225 |
| 18 | 0.220 | 227 |
| 20 | 0.223 | 229 |
| 13 | 0.223 | 231 |
| 14 | 0.227 | 237 |
| 10 | 0.226 | 254 |
| mean | $0.214 \pm 0.007$ | $223 \pm 10.7$ |
| ensemble | 0.160 | 200 |
| nohidden | 0.0715 | 169 |

Table 3: Histogram of the networks' performance by number of misclassified training exemplars.

| error count | networks |
|-------------|----------|
| 0 | |
| 1 | |
| 2 | |
| 3 | * |
| 4 | ** |
| 5 | ****** |
| 6 | *** |
| 7 | ** |
| 8 | * |
| 9 | * |
| 10 | |
| 11 | |
| 12 | |
| 13 | * |
| 14 | |
| 15 | |
| 16 | |

# References

[1] J. Hampshire and A. Waibel. A novel objective function for improved phoneme recognition using time delay neural networks. Technical Report CMU-CS-89-118, Carnegie Mellon University School of Computer Science, March 1989.

[2] Geoffrey E. Hinton, Terrence J. Sejnowski, and David H. Ackley. Boltzmann Machines: Constraint satisfaction networks that learn. Technical Report CMU-CS-84-119, Carnegie-Mellon University, May 1984.

[3] Nathan Intrator. A neural network for feature extraction. In D. S. Touretzky, editor, *Advances in Neural Information Processing Systems 2*, pages 719–726, San Mateo, CA, 1990. Morgan Kaufmann.

[4] Willian P. Lincoln and Josef Skrzypek. Synergy of clustering multiple back propagation networks. In D. S. Touretzky, editor, *Advances in Neural Information Processing Systems 2*, pages 650–657, San Mateo, CA, 1990. Morgan Kaufmann.

[5] L. G. Valiant. A theory of the learnable. *Communications of the ACM*, 27(11):1134–1142, 1984.
